# DiscLDA: Discriminative Learning for Dimensionality Reduction and Classification

**Simon Lacoste-Julien**
Computer Science Division
UC Berkeley
Berkeley, CA 94720

**Fei Sha**
Dept. of Computer Science
University of Southern California
Los Angeles, CA 90089

**Michael I. Jordan**
Dept. of EECS and Statistics
UC Berkeley
Berkeley, CA 94720

## Abstract

Probabilistic topic models have become popular as methods for dimensionality reduction in collections of text documents or images. These models are usually treated as generative models and trained using maximum likelihood or Bayesian methods. In this paper, we discuss an alternative: a discriminative framework in which we assume that supervised side information is present, and in which we wish to take that side information into account in finding a reduced dimensionality representation. Specifically, we present DiscLDA, a discriminative variation on Latent Dirichlet Allocation (LDA) in which a class-dependent linear transformation is introduced on the topic mixture proportions. This parameter is estimated by maximizing the conditional likelihood. By using the transformed topic mixture proportions as a new representation of documents, we obtain a supervised dimensionality reduction algorithm that uncovers the latent structure in a document collection while preserving predictive power for the task of classification. We compare the predictive power of the latent structure of DiscLDA with unsupervised LDA on the 20 Newsgroups document classification task and show how our model can identify shared topics across classes as well as class-dependent topics.

## 1 Introduction

Dimensionality reduction is a common and often necessary step in most machine learning applications and high-dimensional data analyses. There is a rich history and literature on the subject, ranging from classical linear methods such as principal component analysis (PCA) and Fisher discriminant analysis (FDA) to a variety of nonlinear procedures such as kernelized versions of PCA and FDA as well as manifold learning algorithms.

A recent trend in dimensionality reduction is to focus on probabilistic models. These models, which include generative topological mapping, factor analysis, independent component analysis and probabilistic latent semantic analysis (pLSA), are generally specified in terms of an underlying independence assumption or low-rank assumption. The models are generally fit with maximum likelihood, although Bayesian methods are sometimes used. In particular, Latent Dirichlet Allocation (LDA) is a Bayesian model in the spirit of pLSA that models each data point (e.g., a document) as a collection of draws from a mixture model in which each mixture component is known as a *topic* [3]. The mixing proportions across topics are document-specific, and the posterior distribution across these mixing proportions provides a reduced representation of the document. This model has been used successfully in a number of applied domains, including information retrieval, vision and bioinformatics [8, 1].

The dimensionality reduction methods that we have discussed thus far are entirely unsupervised. Another branch of research, known as *sufficient dimension reduction* (SDR), aims at making use of

supervisory data in dimension reduction [4, 7]. For example, we may have class labels or regression responses at our disposal. The goal of SDR is then to identify a subspace or other low-dimensional object that retains as much information as possible about the supervisory signal. Having reduced dimensionality in this way, one may wish to subsequently build a classifier or regressor in the reduced representation. But there are other goals for the dimension reduction as well, including visualization, domain understanding, and domain transfer (i.e., predicting a different set of labels or responses).

In this paper, we aim to combine these two lines of research and consider a supervised form of LDA. In particular, we wish to incorporate side information such as class labels into LDA, while retaining its favorable unsupervised dimensionality reduction abilities. The goal is to develop parameter estimation procedures that yield LDA topics that characterize the corpus and maximally exploit the predictive power of the side information.

As a parametric generative model, parameters in LDA are typically estimated with maximum likelihood estimation or Bayesian posterior inference. Such estimates are not necessarily optimal for yielding representations for prediction and regression. In this paper, we use a discriminative learning criterion—conditional likelihood—to train a variant of the LDA model. Moreover, we augment the LDA parameterization by introducing class-label-dependent auxiliary parameters that can be tuned by the discriminative criterion. By retaining the original LDA parameters and introducing these auxiliary parameters, we are able to retain the advantages of the likelihood-based training procedure and provide additional freedom for tracking the side information.

The paper is organized as follows. In Section 2, we introduce the discriminatively trained LDA (DiscLDA) model and contrast it to other related variants of LDA models. In Section 3, we describe our approach to parameter estimation for the DiscLDA model. In Section 4, we report empirical results on applying DiscLDA to model text documents. Finally, in Section 5 we present our conclusions.

## 2 Model

We start by reviewing the LDA model [3] for topic modeling. We then describe our extension to LDA that incorporates class-dependent auxiliary parameters. These parameters are to be estimated based on supervised information provided in the training data set.

### 2.1 LDA

The LDA model is a generative process where each document in the text corpus is modeled as a set of draws from a mixture distribution over a set of hidden topics. A topic is modeled as a probability distribution over words. Let the vector $\boldsymbol{w}_d$ be the bag-of-words representation of document $d$. The generative process for this vector is illustrated in Fig. 1 and has three steps: 1) the document is first associated with a $K$-dimensional topic mixing vector $\boldsymbol{\theta}_d$ which is drawn from a Dirichlet distribution, $\boldsymbol{\theta}_d \sim \mathrm{Dir}(\alpha)$; 2) each word $w_{dn}$ in the document is then assigned to a single topic $z_{dn}$ drawn from the multinomial variable, $z_{dn} \sim \mathrm{Multi}(\boldsymbol{\theta}_d)$; 3) finally, the word $w_{dn}$ is drawn from a $V$-dimensional multinomial variable, $w_{dn} \sim \mathrm{Multi}(\boldsymbol{\phi}_{z_{dn}})$, where $V$ is the size of the vocabulary. Given a set of documents, $\{\boldsymbol{w}_d\}_{d=1}^D$, the principal task is to estimate the parameters $\{\boldsymbol{\phi}_k\}_{k=1}^K$. This can be done by maximum likelihood, $\boldsymbol{\Phi}^* = \arg\max_{\boldsymbol{\Phi}} p(\{\boldsymbol{w}_d\}; \boldsymbol{\Phi})$, where $\boldsymbol{\Phi} \in \Re^{V \times K}$ is a matrix parameter whose columns $\{\boldsymbol{\phi}_k\}_{k=1}^K$ are constrained to be members of a probability simplex. It is also possible to place a prior probability distribution on the word probability vectors $\{\boldsymbol{\phi}_k\}_{k=1}^K$—e.g., a Dirichlet prior, $\boldsymbol{\phi}_k \sim \mathrm{Dir}(\beta)$—and treat the parameter $\boldsymbol{\Phi}$ as well as the hyperparameters $\alpha$ and $\beta$ via Bayesian methods. In both the maximum likelihood and Bayesian framework it is necessary to integrate over $\boldsymbol{\theta}_d$ to obtain the marginal likelihood, and this is accomplished either using variational inference or Gibbs sampling [3, 8].

### 2.2 DiscLDA

In our setting, each document is additionally associated with a categorical variable or class label $y_d \in \{1, 2, \ldots, C\}$ (encoding, for example, whether a message was posted in the newsgroup alt.atheism vs. talk.religion.misc). To model this labeling information, we introduce a simple extension to the standard LDA model. Specifically, for each class label $y$, we introduce a linear transformation $\boldsymbol{T}^y : \Re^K \to \Re_+^L$, which transforms a $K$-dimensional Dirichlet variable $\boldsymbol{\theta}_d$ to

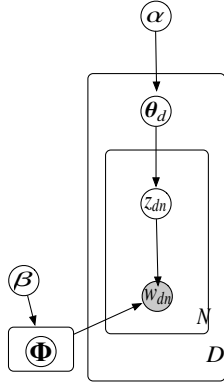

Figure 1: LDA model.

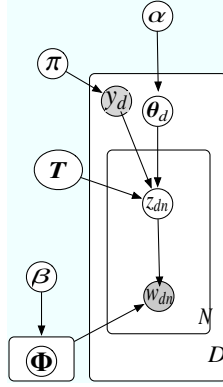

Figure 2: DiscLDA.

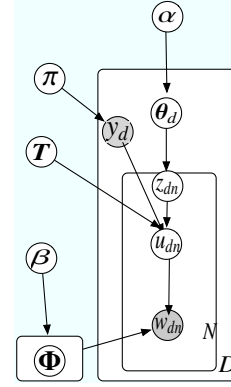

Figure 3: DiscLDA with auxiliary variable $u$.

a mixture of Dirichlet distributions: $\boldsymbol{T}^y \boldsymbol{\theta}_d \in \Re^L$. To generate a word $w_{dn}$, we draw its topic $z_{dn}$ from $\boldsymbol{T}^{y_d} \boldsymbol{\theta}_d$. Note that $\boldsymbol{T}^y$ is constrained to have its columns sum to one to ensure the normalization of the transformed variable $\boldsymbol{T}^y \boldsymbol{\theta}_d$ and is thus a stochastic matrix. Intuitively, every document in the text corpus is represented through $\boldsymbol{\theta}_d$ as a point in the topic simplex $\{\boldsymbol{\theta} \mid \sum_k \theta_k = 1\}$, and we hope that the linear transformation $\{\boldsymbol{T}^y\}$ will be able to *reposition* these points such that documents with the same class labels are represented by points nearby to each other. Note that these points can *not* be placed arbitrarily, as all documents—whether they have the same class labels or they do not— share the parameter $\boldsymbol{\Phi} \in \Re^{V \times L}$. The graphical model in Figure 2 shows the new generative process. Compared to standard LDA, we have added the nodes for the variable $y_d$ (and its prior distribution $\pi$), the transformation matrices $\boldsymbol{T}^y$ and the corresponding edges.

An alternative to DiscLDA would be a model in which there are class-dependent topic parameters $\phi_k^y$ which determine the conditional distribution of the words:

$$w_{dn} \mid z_{dn}, y_d, \boldsymbol{\Phi} \sim \text{Multi}(\boldsymbol{\phi}_{z_{dn}}^{y_d}).$$

The problem with this approach is that the posterior $p(y|\boldsymbol{w}, \boldsymbol{\Phi})$ is a highly non-convex function of $\boldsymbol{\Phi}$ which makes its optimization very challenging given the high dimensionality of the parameter space in typical applications. Our approach circumvents this difficulty by learning a low-dimensional *transformation* of the $\phi_k$'s in a discriminative manner instead. Indeed, transforming the topic mixture vector $\boldsymbol{\theta}$ is actually equivalent to transforming the $\boldsymbol{\Phi}$ matrix. To see this, note that by marginalizing out the hidden topic vector $\boldsymbol{z}$, we get the following distribution for the word $w_{dn}$ given $\boldsymbol{\theta}$:

$$w_{dn} \mid y_d, \boldsymbol{\theta}_d, \boldsymbol{T} \sim \text{Mult}\left(\boldsymbol{\Phi}\boldsymbol{T}^y\boldsymbol{\theta}_d\right).$$

By the associativity of the matrix product, we see that we obtain an equivalent probabilistic model by applying the linear transformation to $\boldsymbol{\Phi}$ instead, and, in effect, defining the class-dependent topic parameters as follows:

$$\phi_k^y = \sum_l \phi_l T_{lk}^y.$$

Another motivation for our approach is that it gives the model the ability to distinguish topics which are shared across different classes versus topics which are class-specific. For example, this separation can be accomplished by using the following transformations (for binary classification):

$$\boldsymbol{T}^1 = \begin{pmatrix} \boldsymbol{I}_K & \boldsymbol{0} \\ \boldsymbol{0} & \boldsymbol{0} \\ \boldsymbol{0} & \boldsymbol{I}_K \end{pmatrix}, \qquad \boldsymbol{T}^2 = \begin{pmatrix} \boldsymbol{0} & \boldsymbol{0} \\ \boldsymbol{I}_K & \boldsymbol{0} \\ \boldsymbol{0} & \boldsymbol{I}_K \end{pmatrix} \tag{1}$$

where $\boldsymbol{I}_K$ stands for the identity matrix with $K$ rows and columns. In this case, the last $K$ topics are shared by both classes, whereas the two first groups of $K$ topics are exclusive to one class or the other. We will explore this parametric structure later in our experiments.

Note that we can give a generative interpretation to the transformation by augmenting the model with a hidden topic vector variable $\boldsymbol{u}$, as shown in Fig. 3, where

$$p(u = k|z = l, \boldsymbol{T}, y) = T_{kl}^y.$$

In this augmented model $\boldsymbol{T}$ can be interpreted as the probability transition matrix from $\boldsymbol{z}$-topics to $\boldsymbol{u}$-topics.

By including a Dirichlet prior on the $\boldsymbol{T}$ parameters, the DiscLDA model can be related to the author-topic model [10], if we restrict to the special case in which there is only one author per document. In the author-topic model, the bag-of-words representation of a document is augmented by a list of the authors of the document. To generate a word in a document, one first picks at random the author associated with this document. Given the author ($y$ in our notation), a topic is chosen according to *corpus-wide* author-specific topic-mixture proportions (which is a column vector $\boldsymbol{T}^y$ in our notation). The word is then generated from the corresponding topic distribution as usual. According to this analogy, we see that our model not only enables us to predict the author of a document (assuming a small set of possible authors), but we also capture the content of documents (using $\boldsymbol{\theta}$) as well as the corpus-wide class properties (using $\boldsymbol{T}$). The focus of the author-topic model was to model the interests of authors, not the content of documents, explaining why there was no need to add document-specific topic-mixture proportions. Because we want to predict the class for a *specific* document, it is crucial that we also model the content of a document.

Recently, there has been growing interest in topic modeling with supervised information. Blei and McAuliffe [2] proposed a supervised LDA model where the empirical topic vector $\boldsymbol{z}$ (sampled from $\boldsymbol{\theta}$) is used as a covariate for a regression on $y$ (see also [6]). Mimno and McCallum [9] proposed a Dirichlet-multinomial regression which can handle various types of side information, including the case in which this side information is an indicator variable of the class $(y)$[1]. Our work differs from theirs, however, in that we train the transformation parameter by maximum conditional likelihood instead of a generative criterion.

## 3 Inference and learning

Given a corpus of documents and their labels, we estimate the parameters $\{\boldsymbol{T}^y\}$ by maximizing the conditional likelihood $\sum_d \log p(y_d \,|\, \boldsymbol{w}_d; \{\boldsymbol{T}^y\}, \boldsymbol{\Phi})$ while holding $\boldsymbol{\Phi}$ fixed. To estimate the parameters $\boldsymbol{\Phi}$, we hold the transformation matrices fixed and maximize the posterior of the model, in much the same way as in standard LDA models. Intuitively, the two different training objectives have two effects on the model: the optimization of the posterior with respect to $\boldsymbol{\Phi}$ captures the topic structure that is shared in documents throughout a corpus, while the optimization of the conditional likelihood with respect to $\{\boldsymbol{T}^y\}$ finds a transformation of the topics that discriminates between the different classes within the corpus.

We use the Rao-Blackwellized version of Gibbs sampling presented in [8] to obtain samples of $\boldsymbol{z}$ and $\boldsymbol{u}$ with $\boldsymbol{\Phi}$ and $\boldsymbol{\theta}$ marginalized out. Those samples can be used to estimate the likelihood of $p(\boldsymbol{w}|y, \boldsymbol{T})$, and thus the posterior $p(y|\boldsymbol{w}, \boldsymbol{T})$ for prediction, by using the harmonic mean estimator [8]. Even though this estimator can be unstable in general model selection problems, we found that it gave reasonably stable estimates for our purposes.

We maximize the conditional likelihood objective with respect to $\boldsymbol{T}$ by using gradient ascent, for a fixed $\boldsymbol{\Phi}$. The gradient can be estimated by Monte Carlo EM, with samples from the Gibbs sampler. More specifically, we use the matching property of gradients in EM to write the gradient as:

$$\frac{\partial}{\partial \boldsymbol{T}} \log p(\boldsymbol{y}|\boldsymbol{w}, \boldsymbol{T}, \boldsymbol{\Phi}) = \mathbb{E}_{q_t^y(z)}\left[\frac{\partial}{\partial \boldsymbol{T}} \log p(\boldsymbol{w}, \boldsymbol{z}|\boldsymbol{y}, \boldsymbol{T}, \boldsymbol{\Phi})\right] - \mathbb{E}_{r_t(z)}\left[\frac{\partial}{\partial \boldsymbol{T}} \log p(\boldsymbol{w}, \boldsymbol{z}|\boldsymbol{T}, \boldsymbol{\Phi})\right], \quad (2)$$

where $q_t^y(z) = p(\boldsymbol{z}|\boldsymbol{w}, \boldsymbol{y}, \boldsymbol{T}_t, \boldsymbol{\Phi})$, $r_t(z) = p(\boldsymbol{z}|\boldsymbol{w}, \boldsymbol{T}_t, \boldsymbol{\Phi})$ and the derivatives are evaluated at $\boldsymbol{T} = \boldsymbol{T}_t$. We can approximate those expectations using the relevant Gibbs samples. After a few gradient updates, we refit $\boldsymbol{\Phi}$ by its MAP estimate from Gibbs samples.

### 3.1 Dimensionality reduction

We can obtain a supervised dimensionality reduction method by using the average transformed topic vector as the reduced representation of a test document. We estimate it using $\mathbb{E}\left[\boldsymbol{T}^y\boldsymbol{\theta}|\boldsymbol{\Phi}, \boldsymbol{w}, \boldsymbol{T}\right] = \sum_y p(y|\boldsymbol{\Phi}, \boldsymbol{w}, \boldsymbol{T})\mathbb{E}\left[\boldsymbol{T}^y\boldsymbol{\theta}|y, \boldsymbol{\Phi}, \boldsymbol{w}, \boldsymbol{T}\right]$. The first term on the right-hand side of this equation can

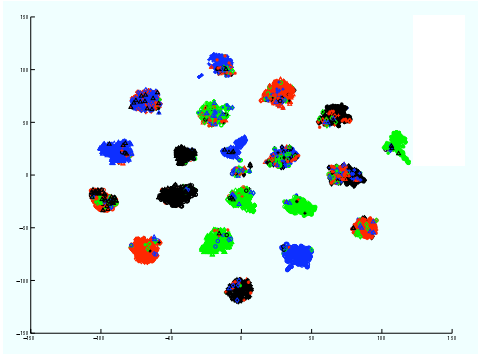 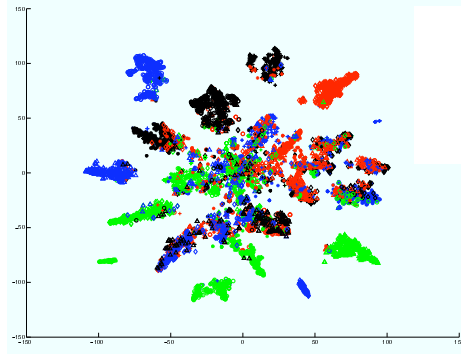

Figure 4: t-SNE $2D$ embedding of the $\mathbb{E}\left[\boldsymbol{T}^y\boldsymbol{\theta}|\boldsymbol{\Phi},\boldsymbol{w},\boldsymbol{T}\right]$ representation of Newsgroups documents, after fitting to the DiscLDA model ($\boldsymbol{T}$ was fixed).

Figure 5: t-SNE $2D$ embedding of the $\mathbb{E}\left[\boldsymbol{\theta}|\boldsymbol{\Phi},\boldsymbol{w},\boldsymbol{T}\right]$ representation of Newsgroups documents, after fitting to the standard unsupervised LDA model.

be estimated using the harmonic mean estimator and the second term can be approximated from MCMC samples of $\boldsymbol{z}$. This new representation can be used as a feature vector for another classifier or for visualization purposes.

## 4   Experimental results

We evaluated the DiscLDA model empirically on text modeling and classification tasks. Our experiments aimed to demonstrate the benefits of discriminative training of LDA for discovering a compact latent representation that contains both predictive and shared components across different types of data. We evaluated the performance of our model by contrasting it to standard LDA models that were not trained discriminatively.

### 4.1   Text modeling

The *20 Newsgroups* dataset contains postings to Usenet newsgroups. The postings are organized by content into 20 related categories and are therefore well suited for topic modeling. In this section, we investigate how DiscLDA can exploit the labeling information—the category—in discovering meaningful hidden structures that differ from those found using unsupervised techniques.

We fit the dataset to both a standard 110-topic LDA model and a DiscLDA model with restricted forms of the transformation matrices $\{\boldsymbol{T}^y\}_{y=1}^{y=20}$. Specifically, the transformation matrix $\boldsymbol{T}^y$ for class label $c$ is fixed and given by the following blocked matrix

$$\boldsymbol{T}^y = \begin{pmatrix} \mathbf{0} & \mathbf{0} \\ \vdots & \vdots \\ \boldsymbol{I}_{K_0} & \mathbf{0} \\ \vdots & \vdots \\ \mathbf{0} & \boldsymbol{I}_{K_1} \end{pmatrix}. \tag{3}$$

This matrix has $(C+1)$ rows and two columns of block matrices. All but two block matrices are zero matrices. At the first column and the row $y$, the block matrix is an identity matrix with dimensionality of $K_0 \times K_0$. The last element of $\boldsymbol{T}^y$ is another identity matrix with dimensionality $K_1$. When applying the transformation to a topic vector $\boldsymbol{\theta} \in \Re^{K_0+K_1}$, we obtain a transformed topic vector $\boldsymbol{\theta}_{\mathrm{tr}} = \boldsymbol{T}^y\boldsymbol{\theta}$ whose nonzeros elements partition the components $\boldsymbol{\theta}_{\mathrm{tr}}$ into $(C+1)$ disjoint sets: one set of $K_0$ elements for each class label that does not overlap with the others, and a set of $K_1$ components that is *shared* by *all* class labels. Intuitively, the shared components should use all class labels to model common latent structures, while nonoverlapping components should model specific characteristics of data from each class.

| Class | Most popular words |
|---|---|
| alt.atheism | atheism, religion, bible, god, system, moral, atheists, keith, jesus, islam, |
| comp.graphics | files, color, images, file, image, format, software, graphics, jpeg, gif, |
| comp.os.ms-windows.misc | card, files, mouse, file, dos, drivers, win, ms, windows, driver, |
| comp.sys.ibmpc.hardware | drive, card, drives, bus, mb, os, disk, scsi, controller, ide, |
| comp.sys.mac.hardware | drive, apple, mac, speed, monitor, mb, quadra, mhz, lc, scsi, |
| comp.windows.x | server, entry, display, file, program, output, window, motif, widget, lib, |
| misc.forsale | price, mail, interested, offer, cover, condition, dos, sale, cd, shipping, |
| rec.autos | cars, price, drive, car, driving, speed, engine, oil, ford, dealer, |
| rec.motorcycles | ca, ride, riding, dog, bmw, helmet, dod, bike, motorcycle, bikes, |
| rec.sport.baseball | games, baseball, year, game, runs, team, hit, players, season, braves, |
| rec.sport.hockey | ca, period, play, games, game, team, win, players, season, hockey, |
| sci.crypt | government, key, public, security, chip, clipper, keys, db, privacy, encryption, |
| sci.electronics | current, power, ground, wire, output, circuit, audio, wiring, voltage, amp, |
| sci.med | gordon, food, disease, pitt, doctor, medical, pain, health, msg, patients, |
| sci.space | earth, space, moon, nasa, orbit, henry, launch, shuttle, satellite, lunar, |
| soc.religion.christian | christians, bible, church, truth, god, faith, christian, christ, jesus, rutgers, |
| talk.politics.guns | people, gun, guns, government, file, fire, fbi, weapons, militia, firearms, |
| talk.politics.mideast | people, turkish, government, jews, israel, israeli, turkey, armenian, armenians, armenia, |
| talk.politics.misc | american, men, war, mr, tax, government, president, health, cramer, stephanopoulos, |
| talk.religion.misc | religion, christians, bible, god, christian, christ, morality, objective, sandvik, jesus, |
| Shared topics | ca, people, post, wrote, group, system, world, work, ll, make, true, university, great, case, number, read, day, mail, information, send, back, article, writes, question, find, things, put, don, cs, didn, good, end, ve, long, point, years, doesn, part, time, state, fact, thing, made, problem, real, david, apr, give, lot, news |

Table 1: Most popular words from each group of class-dependent topics or a bucket of "shared" topics learned in the *20 Newsgroups* experiment with fixed $T$ matrix.

In a first experiment, we examined whether the DiscLDA model can exploit the structure for $T^y$ given in (3). In this experiment, we first obtained an estimate of the $\Phi$ matrix by setting it to the MAP estimate from Gibbs samples as explained in Section 3. We then estimated a new representation for test documents by taking the conditional expectation of $T^y\theta$ with $y$ marginalized out as explained in Section 3.1. Finally, we then computed a 2D-embedding of this $K_1$-dimensional representation of documents. To obtain an embedding, we first tried standard multidimensional scaling (MDS), using the symmetrical $KL$ divergence between pairs of $\theta_{tr}$ topic vectors as a dissimilarity metric, but the results were hard to visualize. A more interpretable embedding was obtained using a modified version of the t-SNE stochastic neighborhood embedding presented by van der Maaten and Hinton [11]. Fig. 4 shows a scatter plot of the $2D$–embedding of the topic representation of the *20 Newsgroups* test documents, where the colors of the dots, each corresponding to a document, encode class labels. Clearly, the documents are well separated in this space. In contrast, the embedding computed from standard LDA, shown in Fig. 5, does not show a clear separation. In this experiment, we have set $K_0 = 5$ and $K_1 = 10$ for DiscLDA, yielding 110 possible topics; hence we set $K = 110$ for the standard LDA model for proper comparison.

It is also instructive to examine in detail the topic structures of the fitted DiscLDA model. Given the specific setup of our transformation matrix $T$, each component of the topic vector $u$ is either associated with a class label or shared across all class labels. For each component, we can compute the most popular words associated from the word-topic distribution $\Phi$. In Table 1, we list these words and group them under each class labels and a special bucket "shared." We see that the words are highly indicative of their associated class labels. Additionally, the words in the "shared" category are "neutral," neither positively nor negatively suggesting proper class labels where they are likely

| LDA+SVM | DiscLDA+SVM | discLDA alone |
|---------|-------------|---------------|
| 20% | 17% | 17% |

Table 2: Binary classification error rates for two newsgroups

to appear. In fact, these words confirm the intuition of the DiscLDA model: they reflect common English usage underlying different documents. We note that we had already taken out a standard list of stop words from the documents.

## 4.2 Document classification

It is also of interest to consider the classification problem more directly and ask whether the features delivered by DiscLDA are more useful for classification than those delivered by LDA. Of course, we can also use DiscLDA as a classification method per se, by marginalizing over the latent variables and computing the probability of the label $y$ given the words in a test document. Our focus in this section, however, is its featural representation. We thus use a different classification method (the SVM) to compare the features obtained by DiscLDA to those obtained from LDA.

In a first experiment, we returned to the fixed $T$ setting studied in Section 4.1 and considered the features obtained by DiscLDA for the *20 Newsgroups* problem. Specifically, we constructed multiclass linear SVM classifiers using the expected topic proportion vectors from unsupervised LDA and DiscLDA models as features as described in Section 3.1. The results were as follows. Using the topic vectors from standard LDA the error rate of classification was $25\%$. When the topic vectors from the DiscLDA model were used we obtained an error rate of $20\%$. Clearly the DiscLDA features have retained information useful for classification.

We also computed the MAP estimate of the class label $y^* = \arg\max p(y|\boldsymbol{w})$ from DiscLDA and used this estimate directly as a classifier. The error rate was again $20\%$.

In a second experiment, we considered the fully adaptive setting in which the transformation matrix $T^y$ is learned in a discriminative fashion as described in Section 3. We initialized the matrix $T$ to a smoothed block diagonal matrix having a pattern similar to (1), with 20 shared topics and 20 class-dependent topics per class. We then sampled $\boldsymbol{u}$ and $\boldsymbol{z}$ for 300 Gibbs steps to obtain an initial estimate of the $\boldsymbol{\Phi}$ vector. This was followed by the discriminative learning process in which we iteratively ran batch gradient (in the log domain, so that $T$ remained normalized) using Monte Carlo EM with a constant step size for 10 epochs. We then re-estimated $\boldsymbol{\Phi}$ by sampling $\boldsymbol{u}$ conditioned on $(\boldsymbol{\Phi}, T)$. This discriminative learning process was repeated until there was no improvement on a validation data set. The step size was chosen by grid search.

In this experiment, we considered the binary classification problem of distinguishing postings of the newsgroup `alt.atheism` from postings of the newsgroup `talk.religion.misc`, a difficult task due to the similarity in content between these two groups.

Table 2 summarizes the results of our experiment, where we have used topic vectors from unsupervised LDA and DiscLDA as input features to binary linear SVM classifiers. We also computed the prediction of the label of a document directly with DiscLDA. As shown in the table, the DiscLDA model clearly generates topic vectors with better predictive power than unsupervised LDA.

In Table 3 we present the ten most probable words for a subset of topics learned using the discriminative DiscLDA approach. We found that the learned $T$ had a block-diagonal structure similar to (3), though differing significantly in some ways. In particular, although we started with 20 shared topics the learned $T$ had only 12 shared topics. We have grouped the topics in Table 3 according to whether they were class-specific or shared, uncovering an interesting latent structure which appears more discriminating than the topics presented in Table 1.

## 5 Discussion

We have presented DiscLDA, a variation on LDA in which the LDA parametrization is augmented to include a transformation matrix and in which this matrix is learned via a conditional likelihood criterion. This approach allows DiscLDA to retain the ability of the LDA approach to find useful

| Topics for alt.atheism | Topics for talk.religion.misc | Shared topics |
|---|---|---|
| god, atheism, religion, atheists, religious, atheist, belief, existence, strong | evil, group, light, read, stop, religions, muslims, understand, excuse | things, bobby, men, makes, bad, mozumder, bill, ultb, isc, rit |
| argument, true, conclusion, fallacy, arguments, valid, form, false, logic, proof | back, gay, convenient, christianity, homosexuality, long, nazis, love, homosexual, david | system, don, moral, morality, murder, natural, isn, claim, order, animals |
| peace, umd, mangoe, god, thing, language, cs, wingate, contradictory, problem | bible, ra, jesus, true, christ, john, issue, church, lds, robert | evidence, truth, statement, simply, accept, claims, explain, science, personal, left |

Table 3: Ten most popular words from a random selection of different types of topics learned in the discriminative learning experiment on the binary dataset.

low-dimensional representations of documents, but to also make use of discriminative side information (labels) in forming these representations.

Although we have focused on LDA, we view our strategy as more broadly useful. A virtue of the probabilistic modeling framework is that it can yield complex models that are modular and can be trained effectively with unsupervised methods. Given the high dimensionality of such models, it may be intractable to train all of the parameters via a discriminative criterion such as conditional likelihood. In this case it may be desirable to pursue a mixed strategy in which we retain the unsupervised criterion for the full parameter space but augment the model with a carefully chosen transformation so as to obtain an auxiliary low-dimensional optimization problem for which conditional likelihood may be more effective.

**Acknowledgements**   We thank the anonymous reviewers as well as Percy Liang, Iain Murray, Guillaume Obozinski and Erik Sudderth for helpful suggestions. Our work was supported by Grant 0509559 from the National Science Foundation and by a grant from Google.

## Footnotes

[1]In this case, their model is actually the same as Model 1 in [5] with an additional prior on the class-dependent parameters for the Dirichlet distribution on the topics.

# References

[1] T. L. Berg, A. C. Berg, J. Edwards, M. Maire, R. White, Y. W. Teh, E. Learned-Miller, and D. A. Forsyth. Names and faces in the news. In *Proceedings of IEEE Conference on Computer Vision and Pattern Recognition*, Washington, DC, 2004.

[2] D. Blei and J. McAuliffe. Supervised topic models. In J. Platt, D. Koller, Y. Singer, and S. Roweis, editors, *Advances in Neural Information Processing Systems 20*, Cambridge, MA, 2008. MIT Press.

[3] D. M. Blei, A. Y. Ng, and M. I. Jordan. Latent Dirichlet allocation. *Journal of Machine Learning Research*, 3:993–1022, 2003.

[4] F. Chiaromonte and R. D. Cook. Sufficient dimension reduction and graphics in regression. *Annals of the Institute of Statistical Mathematics*, 54(4):768–795, 2002.

[5] L. Fei-fei and P. Perona. A Bayesian hierarchical model for learning natural scene categories. In *Proceedings of IEEE Conference on Computer Vision and Pattern Recognition*, San Diego, CA, 2005.

[6] P. Flaherty, G. Giaever, J. Kumm, M. I. Jordan, and A. P. Arkin. A latent variable model for chemogenomic profiling. *Bioinformatics*, 21:3286–3293, 2005.

[7] K. Fukumizu, F. R. Bach, and M. I. Jordan. Kernel dimension reduction in regression. *Annals of Statistics*, 2008. *To appear*.

[8] T. Griffiths and M. Steyvers. Finding scientific topics. *Proceedings of the National Academy of Sciences*, 101:5228–5235, 2004.

[9] D. Mimno and A. McCallum. Topic models conditioned on arbitrary features with Dirichlet-multinomial regression. In *Proceedings of the $24^{th}$ Annual Conference on Uncertainty in Artificial Intelligence*, Helsinki, Finland, 2008.

[10] M. Rosen-Zvi, T. Griffiths T, M. Steyvers, and P. Smyth. The author-topic model for authors and documents. In *Proceedings of the $20^{th}$ Annual Conference on Uncertainty in Artificial Intelligence*, Banff, Canada, 2004.

[11] L. J. P. van der Maaten and G. E. Hinton. Visualizing data using t-SNE. *Journal of Machine Learning Research*, 9:2579–2605, 2008.